# A Holistic Approach to Compositional Semantics: a connectionist model and robot experiments

**Yuuya Sugita**
BSI, RIKEN
Hirosawa 2-1, Wako-shi
Saitama 3510198 JAPAN
*sugita@bdc.brain.riken.go.jp*

Jun Tani
BSI, RIKEN
Hirosawa 2-1, Wako-shi
Saitama 3510198 JAPAN
*tani@bdc.brain.riken.go.jp*

## Abstract

We present a novel connectionist model for acquiring the semantics of a simple language through the behavioral experiences of a real robot. We focus on the "compositionality" of semantics, a fundamental characteristic of human language, which is the ability to understand the meaning of a sentence as a combination of the meanings of words. We also pay much attention to the "embodiment" of a robot, which means that the robot should acquire semantics which matches its body, or sensory-motor system. The essential claim is that an embodied compositional semantic representation can be self-organized from generalized correspondences between sentences and behavioral patterns. This claim is examined and confirmed through simple experiments in which a robot generates corresponding behaviors from unlearned sentences by analogy with the correspondences between learned sentences and behaviors.

## 1  Introduction

Implementing language acquisition systems is one of the most difficult problems, since not only the complexity of the syntactical structure, but also the diversity in the domain of meaning make this problem complicated and intractable. In particular, how linguistic meaning can be represented in the system is crucial, and this problem has been investigated for many years.

In this paper, we introduce a connectionist model to acquire the semantics of language with respect to the behavioral patterns of a real robot. An essential question is how embodied compositional semantics can be acquired in the proposed connectionist model without providing any representations of the meaning of a word or behavior routines *a priori*. By "compositionality", we refer to the fundamental human ability to understand a sentence from (1) the meanings of its constituents, and (2) the way in which they are put together. It is possible for a language acquisition system that acquires compositional semantics to derive the meaning of an unknown sentence from the meanings of known sentences. Consider the unknown sentence: "John likes birds." It could be understood by learning these three sentences: "John likes cats."; "Mary likes birds."; and "Mary likes cats." That is to say, generalization of meaning can be achieved through compositional semantics.

From the point of view of compositionality, the symbolic representation of word meaning has much affinity with processing the linguistic meaning of sentences [4]. Following this observation, various learning models have been proposed to acquire the embodied seman-

tics of language. For example, some models learn semantics in the form of correspondences between sentences and non-linguistic objects, i.e., visual images [10] or the sensory-motor patterns of a robot [7, 13].

In these works, the syntactic aspect of language was acquired through a pre-acquired lexicon. This means that the meanings of words (i.e., lexicon) is acquired independently of the usages of words in sentences (i.e., syntax). Although this separated learning approach seems to be plausible from the requirements of compositionality, it causes inevitable difficulties in representing the meaning of a sentence. A priori separation of lexicon and syntax requires a pre-defined manner of combining word meanings into the meaning of a sentence. In Iwahashi's model, the class of a word is assumed to be given prior to learning its meaning because different acquisition algorithms are required for nouns and verbs (c.f., [12]). Moreover, the meaning of a sentence is obtained by filling a pre-defined template with meanings of words. Roy's model does not require a priori knowledge of word classes, but requires the strong assumption, that the meaning of a word can be assigned to some pre-defined attributes of non-linguistic objects. This assumption is not realistic in more complex cases, such as when the meaning of a word needs to be extracted from non-linguistic spatio-temporal patterns, as in case of learning verbs.

In this paper, we discuss an essential mechanism for self-organizing embodied compositional semantic representations, in which separate treatments of words and syntax are not required. Our model implements compositional semantics by utilizing the generalization capability of an RNN, where the meaning of each word cannot exist independently, but emerges from the relations with others (c.f., reverse compositionality, [3]). In this situation, a sort of generalization can be expected, such that the meanings of novel sentences can be inferred by analogy with learned ones.

The experiments were conducted using a real mobile robot with an arm and with various sensors, including a vision system. A finite set of two-word sentences consisting of a verb followed by a noun was considered. Our analysis will clarify what sorts of internal neural structures should be self-organized for achieving compositional semantics grounded to a robot's behavioral experiences. Although our experimental design is limited, the current study will suggest an essential mechanism for acquiring grounded compositional semantics, with the minimal combinatorial structure of this finite language [2].

## 2 Task Design

The aim of our experimental task is to discuss an essential mechanism for self-organizing compositional semantics based on the behavior of a robot. In the training phase, our robot learns the relationships between sentences and the corresponding behavioral sensory-motor sequences of a robot in a supervised manner. It is then tested to generate behavioral sequences from a given sentence. We regard compositional semantics as being acquired if appropriate behavioral sequences can be generated from unlearned sentences by analogy with learned data.

Our mobile robot has three actuators, with two wheels and a joint on the arm; a colored vision sensor; and two torque sensors, on the wheel and the arm (Figure 1a). The robot operates in an environment where three colored objects (red, blue, and green) are placed on the floor (Figure 1b). The positions of these objects can be varied so long as the robot sees the red object on the left side of its field of view, the green object in the middle, and the blue object on the right at the start of every trial of behavioral sequences. The robot thus learns nine categories of behavioral patterns, consisting of pointing at, pushing, and hitting each of the three objects, in a supervised manner. These categories are denoted as POINT-R, POINT-B, POINT-G, PUSH-R, PUSH-B, PUSH-G, HIT-R, HIT-B, and HIT-G (Figure 1c-e).

The robot also learns sentences which consist of one of 3 verbs (point, push, hit) fol-

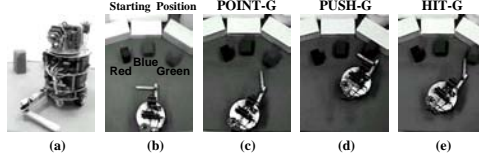

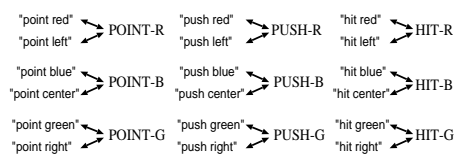

| | Starting Position | POINT-G | PUSH-G | HIT-G |
|---|---|---|---|---|

(a) (b) (c) (d) (e)

Figure 1: The mobile robot (a) starts from a fixed position in the environment and (b) ends each behavior by (c) pointing at, (d) pushing, or (e) hitting an object.

Figure 2: The correspondence between sentences and behavioral categories. Each behavioral category has two corresponding sentences.

lowed by one of 6 nouns (`red`, `left`, `blue`, `center`, `green`, `right`). The meanings of these 18 possible sentences are given in terms of fixed correspondences with the 9 behavioral categories (Figure 2). For example, "`point red`" and "`point left`" correspond to POINT-R, "`point blue`" and "`point center`" to POINT-B, and so on. In these correspondences, "`left`," "`center`," and "`right`" have exactly the same meaning as "`red`," "`blue`," and "`green`" respectively. These synonyms are introduced to observe how the behavioral similarity affects the acquired linguistic semantic structure.

## 3 Proposed Model

Our model employs two RNNs with parametric bias nodes (RNNPBs) [15] in order to implement a linguistic module and a behavioral module (Figure 3). The RNNPB, like the conventional Jordan-type RNN [8], is a connectionist model to learn time sequences. The linguistic module learns the above sentences represented as time sequences of words [1], while the behavioral module learns the behavioral sensory-motor sequences of the robot. To acquire the correspondences between the sentences and behavioral sequences, these two modules are connected to each other by using the parametric bias binding method. Before discussing this binding method in detail, we introduce the overall architecture of RNNPB.

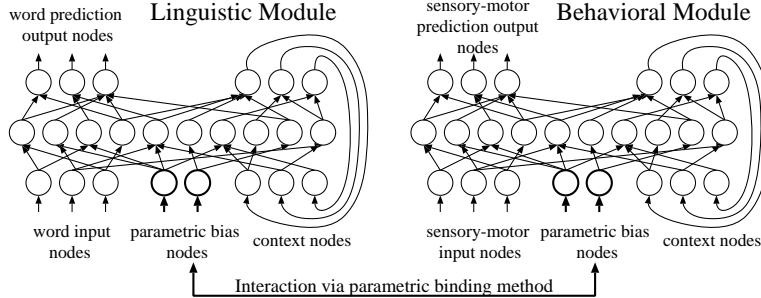

Figure 3: Our model is composed of two RNNs with parametric bias nodes (RNNPBs), one for a linguistic module and the other for a behavioral module. Both modules interact with each other during the learning process via the parametric bias method introduced in the text.

### 3.1 RNNPB

The RNNPB has the same neural architecture as the Jordan-type RNN except for the PB nodes in the input layer (c.f., each module of Figure 3). Unlike the other input nodes, these PB nodes take a specific constant vector throughout each time sequence, and are employed to implement a mapping between fixed-length vectors and time sequences.

Like the conventional Jordan-type RNN, the RNNPB learns time sequences in a supervised manner. The difference is that in the RNNPB, the vectors that encode the time sequences are self-organized in PB nodes during the learning process. The common structural properties of all the training time sequences are acquired as connection weight values by using the

back-propagation through time (BPTT) algorithm, as used also in the conventional RNN [8, 11]. Meanwhile, the specific properties of each individual time sequence are simultaneously encoded as PB vectors (c.f., [9]). As a result, the RNNPB self-organizes a mapping between the PB vectors and the time sequences.

The learning algorithm for the PB vectors is a variant of the BPTT algorithm. For each of $n$ training time sequences of real-numbered vectors $x_0, \cdots, x_{n-1}$, the back-propagated errors with respect to the PB nodes are accumulated for all time steps to update the PB vectors. Formally, the update rule for the PB vector $p_{x_i}$ encoding the $i$-th training time sequence $x_i$ is given as follows:

$$\delta^2 p_{x_i} = \frac{1}{l_i} \sum_{t=0}^{l_i-1} error_{p_{x_i}}(t) \tag{1}$$

$$\delta p_{x_i} = \epsilon \cdot \delta^2 p_{x_i} + \eta \cdot \delta p_{x_i}^{old} \tag{2}$$

$$p_{x_i} = p_{x_i}^{old} + \delta p_{x_i} \tag{3}$$

In equation (1), the update of PB vector $\delta^2 p_{x_i}$ is obtained from the average back-propagated error with respect to a PB node $error_{p_{x_i}}(t)$ through all time steps from $t = 0$ to $l_i - 1$, where $l_i$ is the length of $x_i$. In equation (2), this update is low-pass filtered to inhibit frequent rapid changes in the PB vectors.

After successfully learning the time sequences, the RNNPB can generate a time sequence $x_i$ from its corresponding PB vector $p_{x_i}$. The actual generation process of a time sequence $x_i$ is implemented by iteratively utilizing the RNNPB with the corresponding PB vector $p_{x_i}$, a fixed initial context vector, and input vectors for each time step. Depending on the required functionality, both the external information (e.g., sensory information) and the internal prediction (e.g., motor commands) are employed as input vectors.

Here, we introduce an abstracted operational notation for the RNNPB to facilitate a later explanation of our proposed method of binding language and behavior. By using an operator $RNNPB$, the generation of $x_i$ from $p_{x_i}$ is described as follows:

$$RNNPB(p_{x_i}) \quad \rightarrow \quad x_i, \quad i = 0, \cdots, n - 1. \tag{4}$$

Furthermore, the RNNPB can be used not only for sequence generation processes but also for recognition processes. For a given sequence $x_i$, the corresponding PB vector $p_{x_i}$ can be obtained by using the update rules for the PB vectors (equations (1) to (3)), without updating the connection weight values. This inverse operation for generation is regarded as recognition, and is hence denoted as follows:

$$RNNPB^{-1}(x_i) \quad \rightarrow \quad p_{x_i}, \quad i = 0, \cdots, n - 1. \tag{5}$$

The other important characteristic nature of the RNNPB is that the relational structure among the training time sequences can be acquired in the PB space through the learning process. This generalization capability of RNNPB can be employed to generate and recognize unseen time sequences without any additional learning. For instance, by learning several cyclic time sequences of different frequency, novel time sequences of intermediate frequency can be generated [6].

## 3.2 Binding
In the proposed model, corresponding sentences and behavioral sequences are constrained to have the same PB vectors in both modules. Under this condition, corresponding behavioral sequences can be generated naturally from sentences. When a sentence $s_i$ and its corresponding behavioral sequence $b_i$ have the same PB vector, we can obtain $b_i$ from $s_i$ as follows:

$$RNNPB_B(RNNPB_L^{-1}(s_i)) \quad \rightarrow \quad b_i \tag{6}$$

where $RNNPB_L$ and $RNNPB_B$ are abstracted operators for the linguistic module and the behavioral module, respectively.

The PB vector $p_{s_i}$ is obtained by recognizing the sentence $s_i$. Because of the constraint that corresponding sentences and behavioral sequences must have the same PB vectors, $p_{b_i}$ is equal to $p_{s_i}$. Therefore, we can obtain the corresponding behavioral sequence $\boldsymbol{b}_i$ by utilizing the behavioral module with $p_{b_i}$.

The binding constraint is implemented by introducing an interaction term into part of the update rule for the PB vectors (equation (3)).

$$p_{s_i} = p_{s_i}^{old} + \delta p_{s_i} + \gamma_L \cdot (p_{b_i}^{old} - p_{s_i}^{old}) \tag{7}$$

$$p_{b_i} = p_{b_i}^{old} + \delta p_{b_i} + \gamma_B \cdot (p_{s_i}^{old} - p_{b_i}^{old}) \tag{8}$$

where $\gamma_L$ and $\gamma_B$ are positive coefficients that determine the strength of the binding. Equations (7) and (8) are the constrained update rules for the linguistic module and the behavior module, respectively. Under these rules, the PB vectors of a corresponding sentence $s_i$ and behavioral sequence $\boldsymbol{b}_i$ attract each other. Actually, the corresponding PB vectors $p_{s_i}$ and $p_{b_i}$ need not be completely equalized to learn a correspondence. The epsilon errors of the PB vectors can be neglected because of the continuity of PB spaces.

### 3.3 Generalization of Correspondences

As noted above, our model enables a robot to understand a sentence by means of a generated behavior as if the meaning of the sentence were composed of the meanings of the constituents. That is to say, the robot can generate appropriate behavioral sequences from all sentences without learning all correspondences. To achieve this, an unlearned sentence and its corresponding behavioral sequences must have the same PB vector. Nevertheless, the PB binding method only equalizes the PB vectors for given corresponding sentences and behavioral sequences (c.f., equation (7) and (8)).

Implicit binding, or in other words, inter-module generalization of correspondences, is achieved by dynamic coordination between the PB binding method and the intra-module generalization of each module. The local effect of the PB binding method spreads over the whole PB space, because each individual PB vector depends on the others in order to self-organize PB structures reflecting the relationships among training data. Thus, the PB structures of both modules densely interact via the PB binding methods. Finally, both PB structures converge into a common PB structure, and therefore, all corresponding sentences and behavioral sequences then share the same PB vectors automatically.

## 4 Experiments

In the learning phase, the robot learned 14 of 18 correspondences between sentences and behavioral patterns (c.f., Figure 2). It was then tested to generate behavioral sequences from each of the remaining 4 sentences ("point green", "point right", "push red", and "push left").

To enable a robot to learn correspondences robustly, five corresponding sentences and behavioral sequences were associated by using the PB binding method for each of the 14 training correspondences. Thus, the linguistic module learned 70 sentences with PB binding. Meanwhile, the behavioral module learned the behavioral sequences of the 9 categories, including 2 categories which had no corresponding sentences in the training set. The behavioral module learned 10 different sensory-motor sequences for each behavioral category. It therefore learned 70 behavioral sequences corresponding to the training sentences with PB binding and the remaining 20 sequences independently. In addition, the behavioral module learned the same 90 behavioral sequences without binding.

A sentence is represented as a time sequence of words, which starts with a fixed starting symbol. Each word is locally represented, such that each input node of the module corre-

sponds to a specific word. A single input node takes a value of 1.0 while the others take 0.0 [1]. The linguistic module has 10 input nodes for each of 9 words and a starting symbol. The module also has 6 parametric bias nodes, 4 context nodes, 50 hidden nodes, and 10 prediction output nodes. Thus, no a priori knowledge about the meanings of words is pre-programmed.

A training behavioral sequence was created by sampling three sensory-motor vectors per second during a trial of the robot's human-guided behavior. For robust learning of behavior, each training behavioral sequence was generated under a slightly different environment in which object positions were varied. The variation was at most 20 percent of the distance between the starting position of the robot and the original position of each object in every direction (c.f., Figure 1b). Typical behavioral sequences are about 5 to 25 seconds long, and therefore have about 15 to 75 sensory-motor vectors. A sensory-motor vector is a real-numbered 26-dimensional vector consisting of 3 motor values (for 2 wheels and the arm), 2 values from torque sensors (of the wheels and the arm), and 21 values encoding the visual image. The visual field is divided vertically into 7 regions, and each region is represented by (1) the fraction of the region covered by the object, (2) the dominant hue of the object in the region, and (3) the bottom border of the object in the region, which is proportional to the distance of the object from the camera. The behavioral module had 26 input nodes for sensory-motor input, 6 parametric bias nodes, 6 context nodes, 70 hidden nodes, and 6 output nodes for motor commands and partial prediction of the sensory image at the next time step.

## 5    Results and Analysis

In this section, we analyze the results of the experiment presented in the previous section. The analysis reveals that the inter-module generalization realized by the PB binding method could fill an essential role in self-organizing the compositional semantics of the simple language through the behavioral experiences of the robot. As mentioned in the previous section, the training data for this experiment did not include all the correspondences. As a result, although the behavioral module was trained with the behavioral sequences of all behavioral categories, those in two of the categories, whose corresponding sentences were not in the linguistic training set, could not be bound.

The most important result was that these dangling behavioral sequences could be bound with appropriate sentences. The robot could properly recognize four unseen sentences, and generate the corresponding behaviors. This means that both modules share the common PB structure successfully.

Comparing the PB spaces of both modules shows that they indeed shared a common structure as a result of binding. The linguistic PB vectors are computed by recognizing all the possible 18 sentences including 4 unseen ones (Figure 4a), and the behavioral PB vectors are computed at the learning phase for all the corresponding 90 behavioral sequences in the training data (Figure 4b). The acquired correspondences between sentences and behavioral sequences can be examined according to equation (6). In particular, the implicit binding of the four unlearned correspondences ("`point green`"↔POINT-G, "`point right`"↔POINT-G, "`push red`"↔PUSH-R, and "`push left`"↔PUSH-R) demonstrates acquisition of the underlying semantics, or the generalized correspondences.

The acquired common structure has two striking characteristics: (1) the combinatorial structure originated from the linguistic module, and (2) the metric based on the behavioral similarity originated from the behavioral module. The interaction between modules enabled both PB spaces to simultaneously acquire both of these two structural properties.

We can find three congruent sub-structures for each verb, and six congruent sub-structures for each noun in the linguistic PB space. This congruency represents the underlying syn-

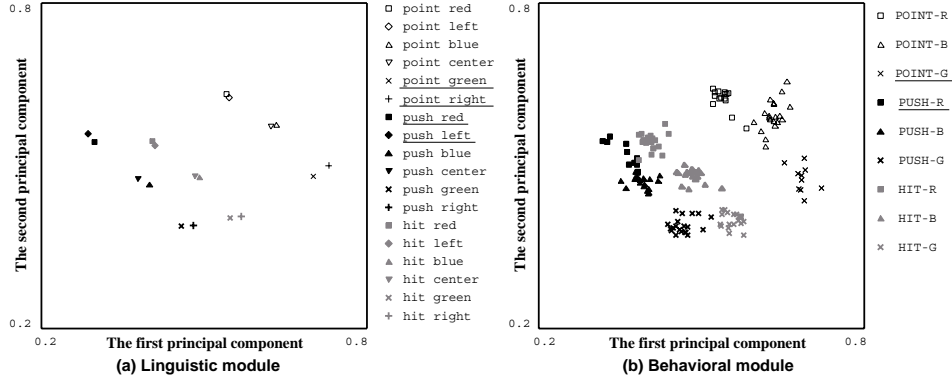

| | |
|---|---|
| □ point red | □ POINT-R |
| ◇ point left | △ POINT-B |
| △ point blue | × POINT-G |
| ▽ point center | ■ PUSH-R |
| × point green | ▲ PUSH-B |
| + point right | × PUSH-G |
| ■ push red | ■ HIT-R |
| ◆ push left | ▲ HIT-B |
| ▲ push blue | × HIT-G |
| ▼ push center | |
| × push green | |
| + push right | |
| ■ hit red | |
| ◆ hit left | |
| ▲ hit blue | |
| ▼ hit center | |
| × hit green | |
| + hit right | |

Figure 4: Plots of the bound linguistic module (a) and the bound behavioral module (b). Both plots are projections of the PB spaces onto the same surface determined by the PCA method. Here, the accumulated contribution rate is about 73%. Unlearned sentences and their corresponding behavioral categories are underlined.

tax structure of training sentences. For example, it is possible to estimate the PB vector of "`point green`" from the relationship among the PB vectors of "`point blue`", "`hit blue`" and "`hit green`." This predictable geometric regularity could be acquired by independent learning of the linguistic module. However it could not be acquired by independent learning of the behavioral module because these behavioral sequences can not be decomposed into plausible primitives, unlike the sentences which can be broken down into words.

We can also see a metric reflecting the similarity of behavioral sequences not only in the behavioral modules but also in the linguistic module. The PB vectors of sentences that correspond to the same behavioral category take the similar values. For example, the two sentences corresponding to POINT-R ("`point red`" and "`point left`") are encoded in similar PB vectors. Such a metric nature could not be observed in the independent learning of the linguistic module, in which all nouns were plotted symmetrically in the PB space by means of the syntactical constraints.

The above observation thus confirms that the embodied compositional semantics was self-organized through the unification of both modules, which was implemented by the PB binding method. We also made experiments with different test sentences, and confirmed that similar results could be obtained.

## 6 Discussion and Summary

Our simple experiments showed that the minimal grounded compositional semantics of our language can be acquired by generalizing the correspondences between sentences and the behavioral sensory-motor sequences of a robot. Our experiments could not examine strong systematicity [4], but could address the combinatorial characteristic nature of sentences. That is to say, the robot could understand relatively simple sentences in a systematic way, and could understand novel sentences. Therefore, our results can elucidate some important issues about the compositional semantic representation.

We claim that the acquisition of word meaning and syntax can not be separated from the standpoint of the symbol grounding problem [5]. The meanings of words depend on each other to compose the meanings of sentences [16]. Consider the meaning of the word "`red`." The meaning of "`red`" must be something which combines with the meaning of "`point`", "`push`" or "`hit`" to form the grounded meanings of sentences. Therefore, a priori definition of the meaning of "`red`" substantially affects the organization of the other parts of the system, and often results in further pre-programming. This means that it is inevitably difficult to explicitly extract the meaning of a word from the meaning of a sentence.

Our model avoids this difficulty by implementing the grounded meaning of a word implicitly in terms of the relationships among the meanings of sentences based on behavioral experiences. Our model does not require any pre-programming of syntactic information, such as symbolic representation of word meaning, a predefined combinatorial structure in the semantic domain, or behavior routines. Instead, the essential structures accounting for compositionality are fully self-organized in the iterative dynamics of the RNN, through the structural interactions between language and behavior using the PB binding method. Thus, the robot can understand "`red`" through its behavioral interactions in the designed tasks in a bottom-up way [14]. A similar argument holds true for verbs. For example, the robot understands "`point`" through pointing at red, blue, and green objects.

To the summary, the current study has shown the importance of generalization of the correspondences between sentences and behavioral patterns in the acquisition of an embodied language. In future studies, we plan to apply our model to larger language sets. In the current experiment, the training set consists of a large fraction of the legal input space, when compared with related works. Such a large training set is needed because our model has no a priori knowledge of syntax and composition rules. However, we think that our model requires relatively fewer fraction of sentences to learn a larger language set, for a given degree of syntactic complexity.

## References

[1] J. L. Elman. Finding structure in time. *Cognitive Science*, 14:179–211, 1990.

[2] G. Evans. Semantic Theory and Tacit Knowledge. In S. Holzman and C. Leich, editors, *Wittgenstein: To Follow a Rule*. London: Routledge and Kegan Paul, 1981.

[3] J. Fodor. Why Compositionality Won't Go Away: Reflections on Horwich's 'Deflationary' Theory. Technical Report 46, Rutgers University, 1999.

[4] R. F. Hadley. Systematicity revisited: reply to Christiansen and Chater and Niklasson and van Gelder. *Mind and Language*, 9:431–444, 1994.

[5] S. Harnad. The symbol grounding problem. *Physica D*, 42:335–346, 1990.

[6] M. Ito and J. Tani. Generalization and Diversity in Dynamic Pattern Learning and Generation by Distributed Representation Architecture . Technical Report 3, Lab. for BDC, Brain Science Institute, RIKEN, 2003.

[7] N. Iwahashi. Language acquisition by robots – Towards New Paradigm of Language Processing –. *Journal of Japanese Society for Artificial Intelligence*, 18(1):49–58, 2003.

[8] M.I. Jordan and D.E. Rumelhart. Forward models: supervised learning with a distal teacher. *Cognitive Science*, 16:307–354, 1992.

[9] R. Miikkulainen. *Subsymbolic Natural Language Processing: An Integrated Model of Script s, Lexicon, and Memory*. MIT Press, 1993.

[10] D. K. Roy. Learning visually grounded words and syntax for a scene description task. *Computer Speech and Language*, 16, 2002.

[11] D. E. Rumelhart, G. E. Hinton, and R. J. Williams. Learning internal representations by error propagation. In D. E. Rumelhart and J. L. Mclelland, editors, *Parallel Distributed Processing*. Cambridge, MA: MIT Press, 1986.

[12] J. M. Siskind. Grounding the Lexical Semantics of Verbs in Visual Perception using Force Dynamics and Event Logic. *Artificial Intelligence Research*, 15:31–90, 2001.

[13] L. Steels. The Emergence of Grammar in Communicating Autonomous Robotic Agents. In W. Horn, editor, *Proceedings of European Conference of Artificial Intelligence*, pages 764–769. IOS Press, 2000.

[14] J. Tani. Model-Based Learning for Mobile Robot Navigation from the Dynamical Systems Perspective. *IEEE Trans. on SMC (B)*, 26(3):421–436, 1996.

[15] J. Tani. Learning to generate articulated behavior through the bottom-up and the top-down interaction process. *Neural Networks*, 16:11–23, 2003.

[16] T. Winograd. Understanding natural language. *Cognitive Psychology*, 3(1):1–191, 1972.
